# Human Active Learning

**Rui Castro**[1]**, Charles Kalish**[2]**, Robert Nowak**[3]**, Ruichen Qian**[4]**, Timothy Rogers**[2]**, Xiaojin Zhu**[4]*

[1]Department of Electrical Engineering
Columbia University. New York, NY 10027
Department of {[2]Psychology, [3]Electrical and Computer Engineering, [4]Computer Sciences}
University of Wisconsin-Madison. Madison, WI 53706

## Abstract

We investigate a topic at the interface of machine learning and cognitive science. Human active learning, where learners can actively query the world for information, is contrasted with passive learning from random examples. Furthermore, we compare human active learning performance with predictions from statistical learning theory. We conduct a series of human category learning experiments inspired by a machine learning task for which active and passive learning error bounds are well understood, and dramatically distinct. Our results indicate that humans are capable of actively selecting informative queries, and in doing so learn better and faster than if they are given random training data, as predicted by learning theory. However, the improvement over passive learning is not as dramatic as that achieved by machine active learning algorithms. To the best of our knowledge, this is the first quantitative study comparing human category learning in active versus passive settings.

## 1 Introduction

Active learning is a paradigm in which the learner has the ability to sequentially select examples for labeling. The selection process can take advantage of information gained from previously observed labeled examples in order to accelerate the learning process. In contrast, passive learning is a paradigm in which the learner has no control over the labeled examples it is given. In machine learning, active learning has been a topic of intense interest. In certain machine learning problems it has been shown that active learning algorithms perform much better than passive learning, with superior convergence bounds (see [1, 4] and references therein) and/or superior empirical performance [5, 19]. In this paper we focus on the application of active learning to classification, in both machines and humans.

To our knowledge, no previous work has attempted to quantify human active learning performance in probabilistic category learning (i.e., classification), contrast human active and passive learning, and compare against theoretically optimal theory bounds. Theories of human category learning often cast the learner as a passive learner, who observes some object (typically represented as a feature vector), is presented with the object's category label, and does some statistical processing to determine how the label should generalize. Anyone who has ever interacted with a three-year-old will recognize that this scenario is exceedingly unrealistic in at least one respect. Certainly toddlers observe their environment, and certainly they pay attention when adults label objects for them – but they also ask a lot of questions. Active querying provides children with information that they would otherwise be less likely to encounter through passive observation; and so, presumably, such active querying has important implications for category learning.

Early research in human concept attainment suggested that learners do benefit from the opportunity to actively select examples during learning [11]. However, it proved very difficult to establish cri-

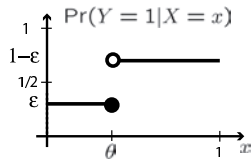

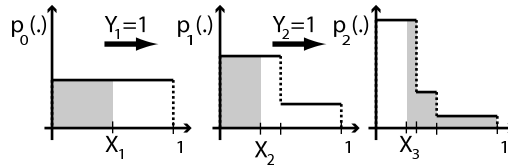

Figure 1: The two-category learning task with boundary $\theta$ and noise level $\epsilon$.

Figure 2: Probabilistic bisection strategy. Shaded areas have $1/2$ probability mass.

teria for assessing the magnitude of the active learning benefit (e.g., compared to theoretical ideals, or to passive learning). Partly as a result, nearly all contemporary research in classification and categorization has ignored active learning. Furthermore, a rich literature on decision-making and scientific inference has produced conflicting claims regarding people's capacities to select optimal learning examples [7, 10, 12, 13, 14, 15, 16, 17, 20]. Most famously, people make inappropriate queries to assess simple logical hypotheses such as "if $p$ then $q$" (frequently examining $q$ instances to see if they are $p$, and failing to explore not-$q$ instances [20]). Several authors have argued that pessimistic views of the human ability to choose relevant queries are based on faulty task analyses; and that, when the learning task is properly construed, humans do an excellent, even optimal job of selection [7, 14]. As much of the debate in the psychological literature turns on task analysis and the proper metric for assessing performance, there is significant opportunity to benefit from the formal descriptions characteristic of machine learning research. The current study exploits one such analysis of a relatively simple binary classification task with fixed error rate in feedback. Specification of the theoretical benefits of active learning in this context allows us to address the following questions regarding human performance:

**[Q1]** Do humans perform better when they can select their own examples for labeling, compared to passive observation of labeled examples?

**[Q2]** If so, do they achieve the full benefit of active learning suggested by statistical learning theory?

**[Q3]** If they do not, can machine learning be used to enhance human performance?

**[Q4]** Do the answers to these questions vary depending upon the difficulty of the learning problem?

The goal of this paper is to answer these questions in a quantitative way by studying human and machine performance in one well-understood classification task. Answers to these questions have important theoretical and practical implications for our understanding of human learning and cognition. As previously noted, most theories of human category learning assume passive sampling of the environment. Some researchers have argued that the environment provides little information regarding the category structure of the world, and so conclude that human category learning must be subject to strong initial constraints [6, 3, 9]. If, however, human learning benefits from active querying of the environment, it is not clear that such conclusions are justified. From an applied perspective, if machines can be shown to aid human learning in certain predictable circumstances, this has clear implications for the design of intelligent tutoring systems and other machine-human hybrid applications.

## 2  A Two-Category Learning Task

For the study in this paper we consider learning in a relatively simple setting, where there is a good theoretical understanding of both active and passive machine learning, offering an ideal test-bed for assessing active learning in humans. The task is essentially a two-category learning problem (binary classification) in the interval $[0, 1]$. Let $\theta \in [0, 1]$ be the unknown but fixed decision boundary. To the left of $\theta$ the category is "zero" and to the right of $\theta$ the category is "one." The goal of the learning task is to infer $\theta$ as accurately as possible from a set of examples. The training data (set of examples) consists of $n$ sample and label pairs; $\{(X_i, Y_i)\}_{i=1}^{n}$, where $X_i \in [0, 1]$ and $Y_i \in \{0, 1\}$. The label $Y_i$ is related to the sample $X_i$ in the following noisy way: $Y_i$ is equal to the category of $X_i$ with probability $1 - \epsilon$ and equal to the other category with probability $\epsilon$, where $0 \leq \epsilon < 1/2$. In other words, each label more probably is correct than incorrect, and $\epsilon$ is the probability of an incorrect

label[1]. Note that the label $Y_i$ is simply a noisy answer to the question "is $X_i$ larger than $\theta$?" Figure 1 illustrates this model. Furthermore assume that, given $X_i$, $Y_i$ is statistically independent of $\{Y_j\}_{j \neq i}$.

At this point we have not specified how the sample locations $X_i$ are generated, and in this lies the major difference between passive and active learning. In the passive learning setting the sample locations are randomly distributed, independent of the labels. On the other hand, in the active learning setting the learner can choose the sample locations in a sequential way depending on the past, that is $X_i = h(X_1, \ldots, X_{i-1}, Y_1, \ldots, Y_{i-1})$, where $h$ is a (possibly random) function that takes into account past experiences and proposes a new query $X_i$.

If $\epsilon = 0$, that is when there is no label noise, the optimal methodologies for passive and active learning are quite obvious. In passive learning, the optimal inference is that $\theta$ lies somewhere between the rightmost location where a label of zero was observed and the leftmost location where a label of one was observed. If the $n$ sample locations are (approximately) evenly distributed between 0 and 1, then the error of the inference is on the order of $1/n$. On the other hand, in active learning the optimal strategy is a deterministic binary bisection: begin by taking $X_1 = 1/2$. If $Y_1 = 0$, then $\theta > 1/2$, otherwise $\theta \leq 1/2$. Suppose $Y_1 = 1$, then the next sample point is $X_2 = 1/4$ and if $Y_2 = 1$, then $\theta < 1/4$ otherwise $\theta \geq 1/4$. Proceeding in this fashion we see that the length of the interval of possible values of $\theta$ is halved at every observation. Therefore after $n$ samples the error of the active learning inference is at most $2^{-(n+1)}$. Clearly active learning, where the error decays exponentially with the number of samples, is much better than passive learning, where the error can decay only polynomially.

If $\epsilon > 0$ there is uncertainty in our label observation process and estimating $\theta$ becomes more delicate. Under passive learning, the maximum likelihood estimator yields the optimal rate of error convergence. Furthermore it is possible to show a performance lower bound that clarifies what is the best possible performance of *any* passive learning algorithm. In particular we have the following result.

$$\inf_{\hat{\theta}_n} \sup_{\theta \in [0,1]} \mathbb{E}[|\hat{\theta}_n - \theta|] \geq \frac{1}{4} \left( \frac{1 + 2\epsilon}{1 - 2\epsilon} \right)^{2\epsilon} \frac{1}{n+1} , \tag{1}$$

where $\hat{\theta}_n$ is the estimate of $\theta$ obtained after $n$ observations, and the infimum is taken over *all* possible passive learning procedures. This is a so-called minimax lower bound, and gives an indication of the best achievable performance of any passive learning algorithm. That is, no passive algorithm can learn more rapidly. This bound can be easily shown using Theorem 2.2 of [18], and the performance of the maximum likelihood estimator is within a constant factor of (1).

For active learning, deterministic bisection cannot be used due to the label noise. Nevertheless active learning is still extremely beneficial in this setting. Horstein [8] proposed a method that is suitable for our purposes. The key idea stems from Bayesian estimation. Suppose that we have a prior probability density function $p_0(\cdot)$ on the unknown parameter $\theta$, namely that $\theta$ is uniformly distributed over the interval $[0, 1]$. To make the exposition clear let us assume $\theta = 1/4$. Like before, we start by making a query at $X_1 = 1/2$. With probability $1 - \epsilon$ we observe the correct label $Y_1 = 1$, and with probability $\epsilon$ we observe the incorrect label $Y_1 = 0$. Suppose $Y_1 = 1$ was observed. Given these facts we can update the posterior density by applying Bayes rule. In this case we obtain $p_1(t|X_1, Y_1) = 2(1 - \epsilon)$ if $t \leq 1/2$, or $2\epsilon$ if $t > 1/2$. The next step is to choose the sample location $X_2$. We choose $X_2$ so that it *bisects* the posterior probability mass, that is, we take $X_2$ such that $\Pr_{t \sim p_1(\cdot)}(t > X_2 | X_1, Y_1) = \Pr_{t \sim p_1(\cdot)}(t < X_2 | X_1, Y_1)$. In other words $X_2$ is just the median of the posterior distribution. We continue iterating this procedure until we have collected $n$ samples. The estimate $\hat{\theta}_n$ is then defined as the median of the final posterior distribution. Figure 2 illustrates the procedure. Note that if $\epsilon = 0$ then this probabilistic bisection is simply the binary bisection described above.

The above algorithm works extremely well in practice, but it is hard to analyze. In [2] a slightly modified method was introduced, which is more amenable to analysis; the major difference involves

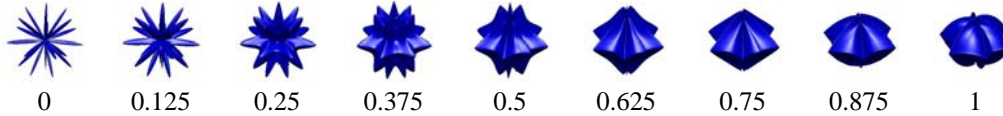

| 0 | 0.125 | 0.25 | 0.375 | 0.5 | 0.625 | 0.75 | 0.875 | 1 |

Figure 3: A few 3D visual stimuli and their $X$ values used in our experiment.

a discretization of the possible query locations. For this method it can be shown [2] that

$$\sup_{\theta \in [0,1]} \mathbb{E}[|\hat{\theta}_n - \theta|] \leq 2 \left( \sqrt{\frac{1}{2} + \sqrt{\epsilon(1-\epsilon)}} \right)^n . \tag{2}$$

Note that the expected estimation error decays exponentially with the number of observations, as opposed to the polynomial decay achievable using passive learning (1). This shows that the accuracy of active learning is significantly better than passive learning, even under the presence of uncertainty. Furthermore no active (or passive) learning algorithm can have their expected error decaying faster than exponentially with the number of samples, as in (2).

## 3  Human Passive and Active Learning Experiments

Equipped with the theoretical performance of passive learning (1) and active learning (2), we now describe a behavioral study designed to answer Q1-Q4 posed earlier. The experiment is essentially a human analog of the abstract learning problem described in the previous section in which the learner tries to find the boundary between two classes defined along a single dimension, a setting used to demonstrate semi-supervised learning behavior in humans in our previous work [21]. We are particularly interested in comparing three distinct conditions:

**Condition "Random"**. This is the passive learning condition where the human subject cannot select the queries, and is instead presented sequentially with examples $\{X_i\}_{i=1}^n$ sampled uniformly at random from $[0,1]$, and their noisy labels $\{Y_i\}_{i=1}^n$. The subject is regularly asked to guess the boundary from these observations (without feedback). As in (1), the expected estimation error $|\hat{\theta}_n - \theta|$ of an optimal machine learning algorithm decreases at the rate $1/n$. If humans are capable of learning from passive observation of random samples, their boundary estimates should approach the true boundary with this polynomial rate too.

**Condition "Human-Active"**. This is the active learning condition where the human subject, at iteration $i$, selects a query $X_i$ based on her previous queries and their noisy labels $\{(X_j, Y_j)\}_{j=1}^{i-1}$. She then receives a subsequent noisy label $Y_i$. If humans are making good use of previously collected examples by selecting informative queries then the rate of error decrease should be exponential, following (2).

**Condition "Machine-Yoked"**. This is a hybrid human-machine-learning condition in which the human passively observes samples selected by the active learning algorithm in [2], observes the noisy label generated in response to each query, and is regularly asked to guess, without feedback, where the boundary is – as though the machine is teaching the human. It is motivated by question Q3: Can machine learning assist human category learning?

**Materials.** Each sample $X$ is a novel artificial 3D shape displayed to the subject on a computer screen. The shapes change with $X$ smoothly in several aspects simultaneously. Figure 3 shows a few shapes and their $X$ values. A difference of 0.06 in $X$ value corresponds roughly to the psychological "Just Noticeable Difference" determined by a pilot study. For implementation reasons our shapes are discretized to a resolution of about 0.003 in $X$ values, beyond which the visual difference is too small to be of interest.

**Participants.** Participants were 33 university students, participating voluntarily or for partial course credit. They were told that the 3D shapes are alien eggs. Spiky eggs ($X$ close to 0) most likely hatch alien snakes (category zero), and smooth eggs ($X$ close to 1) most likely hatch alien birds (category one), but there could be exceptions (label noise). Their task was to identify as precisely as possible the egg shape (decision boundary) at which it switches from most likely snakes to most likely birds.

**Procedure.** Each participant was assigned one of the three conditions: Random (13 subjects), Human-Active (14 subjects), Machine-Yoked (6 subjects). Machine-Yoked receives approximately half the number of other groups, as pilot studies indicated that performance was much less variable in this condition. In all conditions, subjects were explicitly informed of the one dimensional nature of the task. The participant first completed a short practice session to familiarize her with the computer interface and basic task, followed by 5 longer sessions of 45 iterations each. The noise level $\epsilon$, which determines the difficulty of the learning task, varied across sessions, taking the values 0, 0.05, 0.1, 0.2, 0.4 with order determined randomly for each participant. For each session and participant the true decision boundary $\theta$ was randomly set in $[1/16, 15/16]$ to avoid dependencies on the location of the true boundary. The experiment thus involved one between-subject factor (learning condition) and one within-subjects factor (noise level $\epsilon$).

At iteration $i$ of the learning task, a single shape at $X_i$ was displayed on a CRT monitor at a normal viewing distance. In the Human-Active condition, the participant then used a computer mouse wheel to scroll through the range of shapes. Once the participant found the shape she wished to query ($X_{i+1}$), she clicked a "hatch" button and observed the outcome (bird or snake, corresponding to the noisy label), followed by a "Continue" button to move on to the next query. In the Random and Machine-Yoked conditions, each sample $X_{i+1}$ was generated by the computer with no user intervention, and a short animation was displayed showing shapes smoothly transitioning from $X_i$ to $X_{i+1}$ in order to match the visual experience in the Human-Active condition. Once the transition was completed, the outcome (label) for $X_{i+1}$ was observed, and participants clicked a "Continue" button to observe the next sample and outcome. In all conditions, the computer generated the noisy label $Y_{i+1}$ according to the true boundary $\theta$ and noise level $\epsilon$, and displayed it to the participant with either a snake picture ($Y_{i+1} = 0$) or a bird picture ($Y_{i+1} = 1$). The display was reset to the initial shape after ever 3 queries to ensure that participants paid attention to the precise shape corresponding to their estimate of the boundary location rather than simply searching locally around the current shape (total 15 re-starts over 45 queries; 45 re-starts would be too tedious for the subjects).

The participant was asked to guess the decision boundary ($\hat{\theta}$) after every three iterations. In these "boundary queries," the computer began by displaying the shape at $X = 1/2$, and the participant used the mouse wheel to change the shape until it matched her current best guess about the boundary shape. Once satisfied, she clicked a "submit boundary" button. We thus collect $\hat{\theta}_3, \hat{\theta}_6, \hat{\theta}_9, \ldots, \hat{\theta}_{45}$ for each session. These boundary estimates allowed us to compute mean (across subjects) human estimation errors $|\hat{\theta}_n - \theta|$ for different $n$, under different conditions and different noise levels. We compare these means (i) across the different experimental conditions and (ii) to the theoretical predictions in (1)(2).

## 4   Experimental Results

Figure 4 shows, for each condition and noise level, how every participant's boundary guesses approach the true boundary $\theta$. Qualitatively, human active learning (Human-Active) appears better than passive learning (Random) because the curves are more concentrated around zero. Machine-assisted human learning (Machine-Yoked) seems even better. As the task becomes harder (larger noise $\epsilon$), performance suffers in all conditions, though less so for the Machine-Yoked learners. These conclusions are further supported by our quantitative analysis below.

It is worth noting that the behavior of a few participants stand out in Figure 4. For example, one subject's boundary guesses shift considerably within a session, resulting in a rather zigzagged curve in (Human-Active, $\epsilon = 0.1$). All participants, however, perform relatively well in at least some noise settings, suggesting that they took the experiment seriously. Any strange-looking behavior likely reflect genuine difficulties in the task, and for this reason we have not removed any apparent outliers in the following analyses. We now answer questions Q1–Q4 raised in Section 1.

**[Q1] Do humans perform better when they can actively select samples for labeling compared to passive observation of randomly-selected samples?**
**[A1] Yes – at least for low noise levels. For higher noise the two are similar.**
To support our answer, we show that the human estimation error $|\hat{\theta}_n - \theta|$ is smaller in the Human-Active condition than Random condition. This is plotted in Figure 5, with $\pm 1$ standard error bars. When noise is low, the Human-Active curve is well below the Random curve throughout the session.

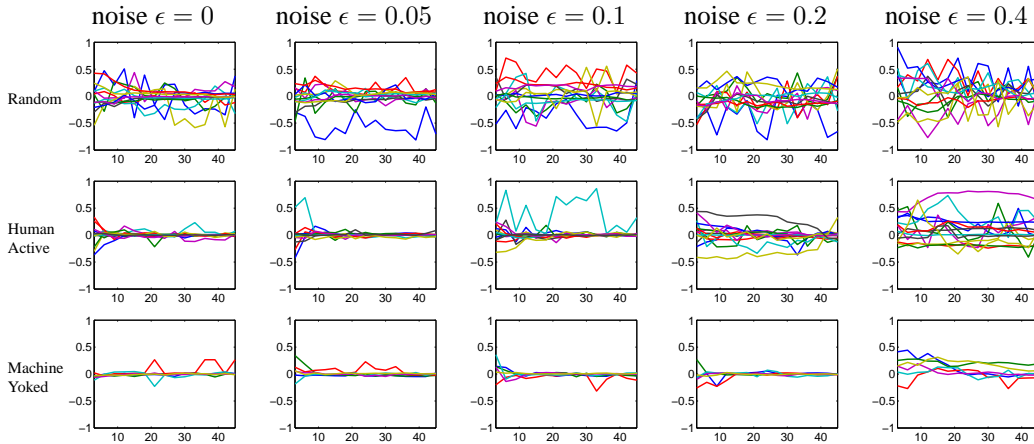

Figure 4: Overview of experiment results. The $x$-axis is iteration $n$, $y$-axis is the (signed) difference between human boundary guess and true boundary $\hat{\theta}_n - \theta$. Each curve shows performance from one human subject (though they overlap, it is sufficient to note the trends). Overall, human active learning (Human-Active) is better than passive learning (Random), and machine-assisted human learning (Machine-Yoked) is even better. As the task becomes harder (larger noise $\epsilon$), all performances suffer.

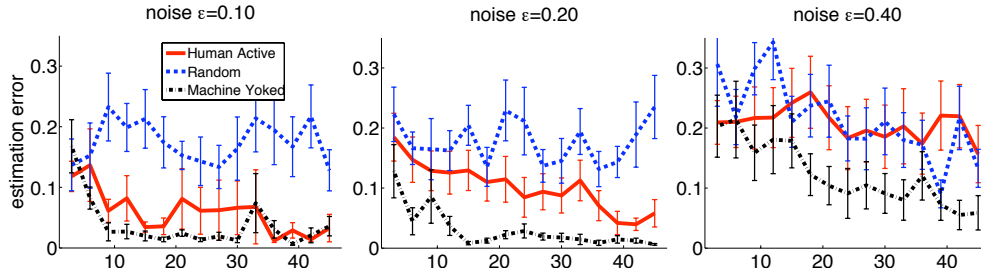

Figure 5: Human estimate error $|\hat{\theta}_n - \theta|$ under different conditions and noise levels. The $x$-axis is iteration $n$. The error bars are $\pm 1$ standard error. Human-Active is better than Random when noise is low; Machine-Yoked is better than Human-Active when noise is high.

That is, with active learning the subjects quickly come up with better guesses and maintain this advantage till the end. Human-Active performance deteriorates with higher noise levels, however, and at the highest noise levels is appears indistinguishable from performance in the Random condition.

**[Q2] Can humans achieve the full benefit of active learning suggested by learning theory?**
**[A2] Human active learning does have exponential convergence, but with slower decay constants than the upper bound in** (2)**. Human passive learning, on the other hand, sometimes does not even achieve polynomial convergence as predicted in** (1)**, and in no condition does the rate approach optimal performance.**

To support these conclusions, consider that, for active learning, the theoretical estimation error bound in (2) has the form $2e^{-\lambda n}$ and decays exponentially with $n$. The decay constant $\lambda = -1/2 \log\left(1/2 + \sqrt{\epsilon(1-\epsilon)}\right)$ is determined by the noise level $\epsilon$. The larger the decay constant, the faster the error approaches zero. If one plots $log$ of the bound vs. $n$, it would be a line with slope $-\lambda$. To determine whether human error decays exponentially as predicted, and with a comparable slope, one can similarly plot the logarithm of *human active learning* estimation error vs. $n$. If human active learning decreases error exponentially (which is desirable), this relationship is linear, as Figure 6 (Upper) shows it to be. This exponential decay of error offers further evidence that human active learning exceeds passive learning performance, where error can only decay polynomially (Figure 6, Lower). The speed (decay constant) of the exponential decay in human active learning is, however, slower than the theoretical upper bound (2). To see this, we fit one line per noise level in

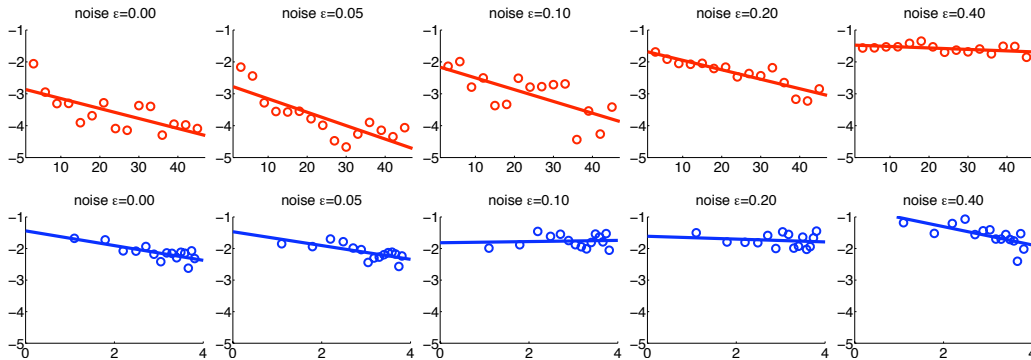

Figure 6: **(Upper)** Human active learning decreases error exponentially, as indicated by the linear distribution of $\log(|\hat{\theta}_n - \theta|)$ (the $y$-axis) versus $n$ (the $x$-axis). **(Lower)** Human passive learning in the Random condition is slower than $O(1/n)$, since the slopes are shallower than -1 on $\log(|\hat{\theta}_n - \theta|)$ (the $y$-axis) versus $\log(n)$ (the $x$-axis).

|  | $\epsilon = 0$ | 0.05 | 0.1 | 0.2 | 0.4 |
|---|---|---|---|---|---|
| Human-Active | 0.031 | 0.042 | 0.037 | 0.030 | 0.005 |
| bound (2) | 0.347 | 0.166 | 0.112 | 0.053 | 0.005 |

Table 1: The exponential decay constants of human active learning is slower than predicted by statistical learning theory for lower noise levels.

Figure 6 and use the negative slope of the fitted lines as the estimate of the decay constant in human active learning. For comparison, we computed the decay constant in the theoretical bound. Table 1 compares these decay constants under different noise levels. It is clear that human active learning's error decays at a slower rate, especially when the noise is low.

For passive learning, the minimax lower bound (1) has a polynomial decay of $O(1/n)$, which is a line with slope -1 on a plot of $\log(|\hat{\theta}_n - \theta|)$ vs. $\log(n)$. As shown in Figure 6 (Lower), the analogous log-log plot from human passive learning in the Random condition does seem to fit a line, but the slope is much shallower than -1. Indeed, for 2 of the 5 noise levels (0.1 and 0.2), the estimated slope is not significantly different from zero! These results suggest that humans either fail to learn or learn at a much lower rate than formal analysis suggests is possible.

**[Q3] Can machine learning be used to enhance human learning?**
**[A3] Apparently in high noise levels – But what really happened?**
As shown in Figure 5, the Machine-Yoked curve is no different than Human-Active in low noise levels, but substantially better in high noise levels. It is important to remember that Machine-Yoked is human performance, not that of the machine learning algorithm. The results seem to indicate that humans can utilize the training data chosen by a machine active learning algorithm to enhance their performance in settings where humans are not generally performing well. Upon closer inspection, however, we noticed that almost all subjects in the Machine-Yoked condition used the following strategy. They quickly learned that the computer was generating training examples that soon converge to the true boundary. They then simply placed their boundary guess at (or near) the latest training example generated by the machine. This "memorizing" strategy worked very well in our setting, but it is difficult to believe that the subjects were really "learning" the decision boundary. Instead, they likely learned to trust and depend upon the computer. In view of this, we consider Q3 inconclusive, but hope these observations provoke thoughts on how to actually improve human learning.

**[Q4] Do answers to the above questions depend upon the difficulty of the learning task?**
**[A4] One form of difficulty, the label noise level $\epsilon$, has profound effects on human learning.**
Specifically, the advantage of active learning diminishes with noise; and at high noise levels active learning arguably has no advantage over passive learning for humans in this setting. Formal analysis

suggests that the advantage of active over passive sampling should diminish with increasing noise; but it also suggests that some benefit to active sampling should always be obtained. An important goal for future research, then, is to understand why human performance is so adversely affected by noise.

## 5 Conclusions and Future Work

We have conducted behavioral experiments to compare active versus passive learning by humans in a simple classification task, and compared human performance to that predicted by statistical learning theory. In short, humans are able to actively select queries and use them to achieve faster category learning; but the advantages of active-learning diminish under higher noise conditions and do not approach theoretical bounds. One important conclusion from this work is that passive learning may not be a very good model for how human beings learn to categorize. Our research also raises several interesting further questions, including how the current conclusions extend to more realistic learning scenarios. The benefit of the current work is that it capitalizes on a simple learning task for which passive and active performance has been formally characterized. The drawback is that the task is not especially natural. In future work we plan to extend the current approach to learning situations more similar to those faced by people in their day-to-day lives.

**Acknowledgments:** This work is supported in part by the Wisconsin Alumni Research Foundation, and NSF Grant 0745423 from Developmental Learning Sciences.

## Footnotes

*Correspondence concerning this article should be send to `jerryzhu@cs.wisc.edu`.

[1]We use a constant noise level $\epsilon$ because the theoretical distinction between active and passive learning is dramatic in this case. Other (perhaps more natural) noise models are possible, for example $\epsilon$ can decrease away from the true class boundary. Noise models like this are well understood theoretically [4]; we will investigate them in future work.

## References

[1] N. Balcan, S. Hanneke, and J. Wortman. The true sample complexity of active learning. *to appear in COLT 2008, Helsinki, Finland*, 2008.

[2] M. V. Burnashev and K. Sh. Zigangirov. An interval estimation problem for controlled observations. *Problems in Information Transmission*, 10:223–231, 1974.

[3] S. Carey. *Conceptual change in childhood*. MIT Press, 1985.

[4] R. Castro and R. Nowak. Minimax bounds for active learning. *IEEE Transactions on Information Theory*, 54(5):2339–2353, 2008.

[5] D. Cohn, L. Atlas, and R. Ladner. Improving generalization with active learning. *Machine Learning*, 15(2):201–221, 1994.

[6] R. Gelman and E. M. Williams. *Handbook of child psychology*, chapter Enabling constraints for cognitive development and learning: A domain-specific epigenetic theory. John Wiley and Sons, 1998.

[7] G. Gigerenzer and R. Selten. *Bounded rationality: The adaptive toolbox*. The MIT Press, 2001.

[8] M. Horstein. Sequential decoding using noiseless feedback. *IEEE Trans. Info. Theory*, 9(3):136–143, 1963.

[9] F. Keil. *Concepts, kinds, and cognitive development*. MIT Press, 1989.

[10] J. K. Kruschke. Bayesian approaches to associative learning: From passive to active learning. *Learning & Behavior*, 36(3):210–226, 2008.

[11] P. A. Laughlin. Focusing strategy in concept attainment as a function of instructions and task complexity. *Journal of Experimental Psychology*, 98(2):320–327, May 1973.

[12] C. R. Mynatt, M. E. Doherty, and R. D. Tweney. Confirmation bias in a simulated research environment: An experimental study of scientific inference. *The Quarterly Journal of Experimental Psychology*, 29(1):85–95, Feb 1977.

[13] J. Nelson. Finding useful questions: On Bayesian diagnosticity, probability, impact, and information gain. *Psychological Review*, 112(4):979–999, 2005.

[14] M. Oaksford and N. Chater. *Bayesian rationality the probabilistic approach to human reasoning*. Oxford University Press, 2007.

[15] L. E. Schulz, T. Kushnir, and A. Gopnik. *Causal Learning; Psychology, Philosophy and Computation*, chapter Learning from doing: Interventions and causal inference. Oxford University Press, 2007.

[16] D. Sobel and T. Kushnir. Interventions do not solely benefit causal learning: Being told what to do results in worse learning than doing it yourself. In *Proceedings of the 25th Annual Meeting of the Cognitive Science Society*, 2003.

[17] M. Steyvers, J. Tenenbaumb, E. Wagenmakers, and B. Blum. Inferring causal networks from observations and interventions. *Cognitive Science*, 27:453–489, 2003.

[18] Alexandre B. Tsybakov. *Introduction à l'estimation non-paramétrique*. Mathématiques et Applications, 41. Springer, 2004.

[19] G. Tur, D. Hakkani-Tür, and R. E. Schapire. Combining active and semi-supervised learning for spoken language understanding. *Speech Communication*, 45:171–186, 2005.

[20] P. C. Wason and P. N. Johnson-Laird. *Psychology of reasoning: Structure and content*. Harvard U. Press, 1972.

[21] X. Zhu, T. Rogers, R. Qian, and C. Kalish. Humans perform semi-supervised classification too. In *Twenty-Second AAAI Conference on Artificial Intelligence*, 2007.

